# Modeling Neural Population Spiking Activity with Gibbs Distributions

**Frank Wood, Stefan Roth, and Michael J. Black**
Department of Computer Science
Brown University
Providence, RI 02912
{fwood,roth,black}@cs.brown.edu

## Abstract

Probabilistic modeling of correlated neural population firing activity is central to understanding the neural code and building practical decoding algorithms. No parametric models currently exist for modeling multivariate correlated neural data and the high dimensional nature of the data makes fully non-parametric methods impractical. To address these problems we propose an energy-based model in which the joint probability of neural activity is represented using learned functions of the 1D marginal histograms of the data. The parameters of the model are learned using contrastive divergence and an optimization procedure for finding appropriate marginal directions. We evaluate the method using real data recorded from a population of motor cortical neurons. In particular, we model the joint probability of population spiking times and 2D hand position and show that the likelihood of test data under our model is significantly higher than under other models. These results suggest that our model captures correlations in the firing activity. Our rich probabilistic model of neural population activity is a step towards both measurement of the importance of correlations in neural coding and improved decoding of population activity.

## 1 Introduction

Modeling population activity is central to many problems in the analysis of neural data. Traditional methods of analysis have used single cells and simple stimuli to make the problems tractable. Current multi-electrode technology, however, allows the activity of tens or hundreds of cells to be recorded simultaneously along with with complex natural stimuli or behavior. Probabilistic modeling of this data is challenging due to its high-dimensional nature and the correlated firing activity of neural populations. One can view the problem as one of learning the joint probability $P(\mathbf{s}, \mathbf{r})$ of a stimulus or behavior $\mathbf{s}$ and the firing activity of a neural population $\mathbf{r}$. The neural activity may be in the form of firing rates or spike times. Here we focus the latter more challenging problem of representing a multivariate probability distribution over spike times.

Modeling $P(\mathbf{s}, \mathbf{r})$ is made challenging by the high dimensional, correlated, and non-Gaussian nature of the data. The dimensionality means that we are unlikely to have suf-

ficient training data for a fully non-parametric model. On the other hand no parametric models currently exist that capture the one-sided, skewed nature of typical correlated neural data. We do, however, have sufficient data to model the marginal statistics of the data. With that observation we draw on the FRAME model developed by Zhu and Mumford for image texture synthesis [1] to represent neural population activity.

The FRAME model represents $P(\mathbf{s}, \mathbf{r})$ in terms of its marginal histograms. In particular we seek the maximum entropy distribution that matches the observed marginals of $P(\mathbf{s}, \mathbf{r})$. The joint is represented by a Gibbs model that combines functions of these marginals and we exploit the method of [2] to automatically choose the optimal marginal directions. To learn the parameters of the model we exploit the technique of contrastive divergence [3, 4] which has been used previously to learn the parameters of Product-of-Experts (PoE) models [5]. We observe that the FRAME model can be viewed as a Product of Experts where the experts are functions of the marginal histograms. The resulting model is more flexible than the standard PoE formulation and allows us to model more complex, skewed distributions observed in neural data.

We train and test the model on real data recorded from a monkey performing a motor control task; details of the task and the neural data are described in the following section. We learn a variety of probabilistic models including full Gaussian, independent Gaussian, product of t-distributions [4], independent non-parametric, and the FRAME model. We evaluate the log likelihood of test data under the different models and show that the complete FRAME model outperforms the other methods (note that "complete" here means the model uses the same number of marginal directions as there are dimensions in the data).

The use of energy-based models such as FRAME for modeling neural data appears novel and promising, and the results reported here are easily extended to other cortical areas. There is a need in the community for such probabilistic models of multi-variate spiking processes. For example Bell and Para [6] formulate a simple model of correlated spiking but acknowledge that what they would really like, and do not have, is what they call a "maximum spikelihood" model. This neural modeling problem represents a new application of energy-based models and consequently suggests extensions of the basic methods. Finally, there is a need for rich probabilistic models of this type in the Bayesian decoding of neural activity [7].

## 2  Methods

The data used in this study consists of simultaneously recorded spike times from a population of M1 motor neurons recorded in monkeys trained to perform a manual tracking task [8, 9]. The monkey viewed a computer monitor displaying a target and a feedback cursor. The task involved moving a 2D manipulandum so that a cursor controlled by the manipulandum came into contact with a target. The monkey was rewarded when the target was acquired, a new target appeared and the process repeated. Several papers [9, 11, 10] have reported successfully decoding the cursor kinematics from this data using firing rates estimated from binned spike counts.

The activity of a population of cells was recorded at a rate of 30kHz then sorted using an automated spike sorting method; from this we randomly selected five cells with which to demonstrate our method.

As shown in Fig. 1, $\mathbf{r}_{i,k} = [t_{i,k}^{(1)}, t_{i,k}^{(2)}, \ldots, t_{i,k}^{(J)}]$ is a vector of time intervals $t_{i,k}^{(j)}$ that represents the spiking activity of a single cell $i$ at timestep $k$. These intervals are the elapsed time between the time at timestep $k$ and the time at each of $j$ past spikes. Let $R_k = [\mathbf{r}_{1,k}, \mathbf{r}_{2,k}, \ldots, \mathbf{r}_{N,k}]$ be a vector concatenation of $N$ such spiking activity representations. Let $\mathbf{s}_k = [x_k, y_k]$ be the position of the manipulandum at each timestep. Our

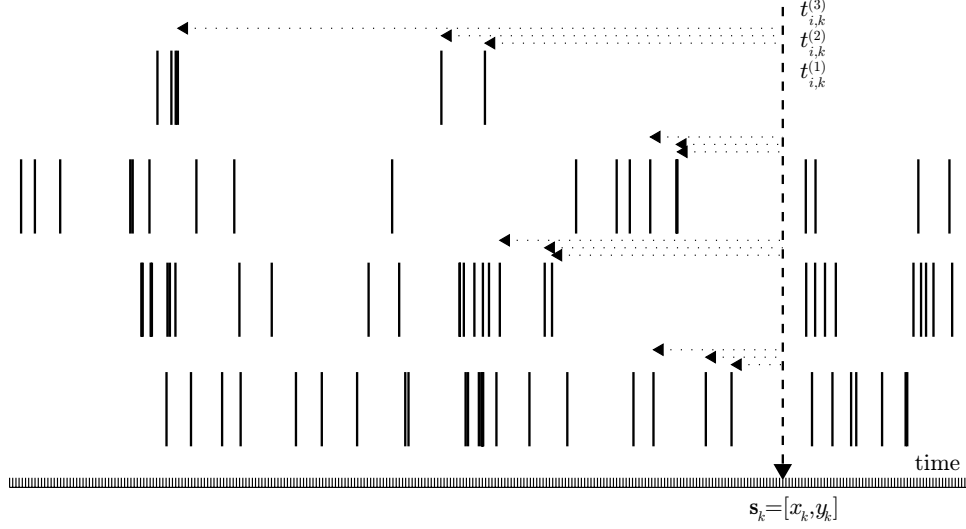

Figure 1: Representation of the data. Hand position at time $k$, $\mathbf{s}_k = [x_k, y_k]$, is regularly sampled every $50ms$. Spiking activity (shown as vertical bars) is retained at full data acquisition precision (30khz). Sections of spike trains from four cells are shown. The response of a single cell, $i$, is represented by the time intervals to the three preceding spikes; that is, $\mathbf{r}_{i,k} = [t_{i,k}^{(1)}, t_{i,k}^{(2)}, t_{i,k}^{(3)}]$.

training data consists of 4000 points $R_k, s_k$ sampled at $50ms$ intervals with a history of 3 past spikes ($J = 3$) per neuron. Our test data is 1000 points of the same.

Various empirical marginals of the data (shown in Fig 2) illustrate that the data are not well fit by canonical symmetric parametric distributions because the data is asymmetric and skewed. For such data traditional parametric models may not work well so instead we apply the FRAME model of [1] to this modeling problem. FRAME is a semi-parametric energy based model of the following form:

Let $\mathbf{d}_k = [\mathbf{s}_k, R_k]$, where $\mathbf{s}_k$ and $R_k$ are defined as above. Let $D = [\mathbf{d}_1, \ldots, \mathbf{d}_N]$ be a matrix of $N$ such points. We define

$$P(\mathbf{d}_k) = \frac{1}{Z(\Theta)} e^{-\sum_e \lambda_e^T \phi(\omega_e^T \mathbf{d}_k)} \tag{1}$$

where $\omega_e$ is a vector that projects the datum $\mathbf{d}_k$ onto a 1-D subspace, $\phi : \mathbb{R} \to \mathbb{I}^b$ is a "histogramming" function that produces a vector with a single 1 in a single bin per datum according to the projected value of that datum, $\lambda_e \in \mathbb{R}^b$ is a weight vector, $Z$ is a normalization constant sometimes called the partition function (as it is a function of the model parameters), $b$ is the granularity of the histogram, and $e$ is the number of "experts". Taken together, $\lambda_e^T \phi(\cdot)$ can be thought of as a discrete representation of a function. In this view $\lambda_e^T \phi(\omega_e^T \mathbf{d}_k)$ is an energy function computed over a projection of the data. Models of this form are constrained maximum entropy models, and in this case by adjusting $\lambda_e$ the model marginal projection onto $\omega_e$ is constrained to be identical (ideally) to the empirical marginal over the same projection. Fig. 3 illustrates the model.

To relate this to current PoE models, if $\lambda_e^T \phi(\cdot)$ were replaced with a log Student-t function then this FRAME model would take the same form as the Product-of-Student-t formulation of [12]. Distributions of this form are called Gibbs or energy-based distributions as $\sum_e \lambda_e^T \phi(\omega_e^T \mathbf{d}_k)$ is analogous to the energy in a Boltzmann distribution. Minimizing the

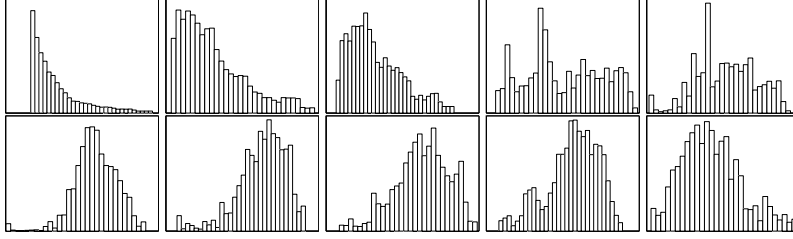

Figure 2: Histograms of various projections of single cell data. The top row are histograms of the values of $t^{(1)}, t^{(2)}, t^{(3)}, x$, and $y$ respectively. The bottom row are random projections from the same data. All these figures illustrate skew or one-sidedness, and motivate our choice of a semi-parametric Gibbs model.

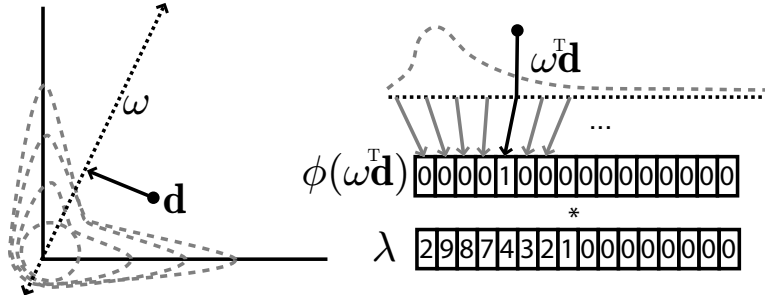

Figure 3: *(left)* Illustration of the projection and weighting of a single point $\mathbf{d}$: Here, the data point $\mathbf{d}$ is projected onto the projection direction $\omega$. The isosurfaces from a hypothetical distribution $p(\mathbf{d})$ are shown in dotted gray. *(right)* Illustration of the projection and binning of $\mathbf{d}$: The upper plot shows the empirical marginal (in dotted gray) as obtained from the projection illustrated in the left figure. The function $\phi(\cdot)$ takes a real valued projection and produces a vector of fixed length with a single 1 in the bin that is mapped to that range of the projection. This discretization of the projection is indicated by the spacing of the downward pointing arrows. The resulting vector is weighted by $\lambda$ to produce an energy. This process is repeated for each of the projection directions in the model. The constraints induced by multiple projections result in a distribution very close to the empirical distribution.

this energy is equivalent to maximizing the log likelihood.

Our model is parameterized by $\Theta = \{\{\lambda_e, \omega_e\} : 1 < e < E\}$ where $E$ is the total number of projections (or "experts"). We use gradient ascent on the log likelihood to train the $\lambda_e$'s. As $\phi(\cdot)$ is not differentiable, the $\omega_e$'s must be specified or learned in another way.

## 2.1 Learning the $\lambda$'s

Standard gradient ascent becomes intractable for large numbers of cells because computing the partition function and its gradient becomes intractable. The gradient of the log probability with respect to $\lambda_{1..E}$ is

$$\nabla_{\Theta_\lambda} \log P(\mathbf{d}_k) = [\frac{\partial \log P(\mathbf{d}_k)}{\partial \lambda_1}, \dots, \frac{\partial \log P(\mathbf{d}_k)}{\partial \lambda_E}]. \tag{2}$$

Besides not being able to normalize the distribution, the right hand term of the partial

$$\frac{\partial \log P(\mathbf{d}_k)}{\partial \lambda_e} = \phi(\omega_e^T \mathbf{d}_k) - \frac{\partial \log Z(\Theta)}{\partial \lambda_e}$$

typically has no closed-form solution and is very hard to compute.

Markov chain Monte Carlo (MCMC) techniques can be used to learn such models. Contrastive divergence [4] is an efficient learning algorithm for energy based models that approximates the gradient as

$$\frac{\partial \log P(\mathbf{d}_k)}{\partial \lambda_e} \approx \left\langle \frac{\partial \log P(\mathbf{d}_k)}{\partial \lambda_e} \right\rangle_{P^0} - \left\langle \frac{\partial \log P(\mathbf{d}_k)}{\partial \lambda_e} \right\rangle_{P_\Theta^m} \tag{3}$$

where $P^0$ is the training data and $P_\Theta^m$ are samples drawn according to the model. The key is that the sampler is started at the training data and does not need to be run until convergence, which typically would take much more time. The superscript indicates that we use $m$ regular Metropolis sampling steps [13] to draw samples from the model for contrastive divergence training ($m = 50$ in our experiments).

The intuition behind this approximation is that samples drawn from the model should have the same statistics as the training data. Maximizing the log probability of training data is equivalent to minimizing the Kullback Leibler (KL) divergence between the model and the true distribution. Contrastive divergence attempts to minimize the difference in KL divergence between the model one step towards equilibrium and the training data. Intuitively this means that the contrastive divergence opposes any tendency for the model to diverge from the true distribution.

## 2.2  Learning the $\omega$'s

Because $\phi(\cdot)$ is not differentiable, we turn to the feature pursuit method of [2] to learn the projection directions $\omega_{1..E}$. This approach involves successively searching for a new projection in a direction where a model with the new projection would differ maximally from the model without. Their approach involves approximating the expected projection using a Parzen window method with Gaussian kernels. Gradient search on a KL-divergence objective function is used to find each subsequent projection. We refer readers to [2] for details.

It was suggested by [2] that there are many local optima in this feature pursuit. Our experience tends to support this claim. In fact, it may be that feature pursuit is not entirely necessary. Additionally, in our experience, the most important aspect of the feature selection algorithm is how many feature pursuit starting points are considered. It may be as effective (and certainly more efficient) to simply guess a large number of projections and estimate the marginal KL-divergence for them all, selecting the largest as the new projection.

## 2.3  Normalizing the distribution

Generally speaking, the partition function is intractible to compute as it involves integration over the entire domain of the joint; however, in the case where $E$ (the number of experts) is the same as the dimensionality of $d$ then the partition function is tractable. Each expert can be normalized individually. The per-expert normalization is

$$Z_e = \sum_b s_e^{(b)} e^{-\lambda_e^{(b)}}$$

where $b$ indexes the elements of $\lambda_e$ and $s_e^{(b)}$ is the width of the $b^{th}$ bin of the $e^{th}$ histogramming function. Using the change of variables rule

$$Z = |det(\Omega)| \prod_e Z_e$$

where the square matrix $\Omega = [\omega_1 \omega_2 \ldots \omega_E]$. This is not possible when the number of experts exceeds or is smaller than the dimensionality of the data.

| POT | IG | G | RF | I | FP |
|-----|-----|-----|-----|-----|-----|
| -31849 | -30893 | -23573 | -23108 | -19155 | -12509 |

Table 1: Log likelihoods of test data. The test data consists of the spiking activity of 5 cells and $x, y$ position behavioral variables as illustrated in Fig. 1. Log likelihoods are reported for various models: POT: Product of Student-t, IG: diagonal covariance Gaussian, G: full covariance Gaussian, RF: random filter FRAME, I: 5 independent FRAME models, one per cell, and FP: feature pursuit FRAME

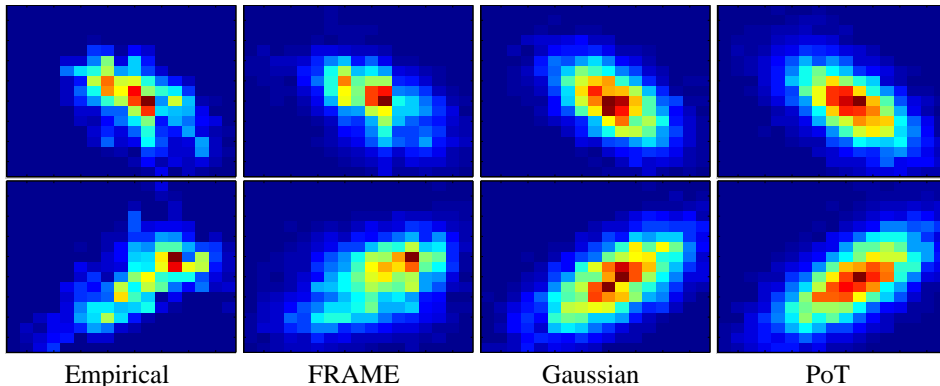

| Empirical | FRAME | Gaussian | PoT |

Figure 4: This figure illustrates the modeling power of the semi-parametric Gibbs distribution over a number of symmetric, fully parametric distributions. Each row shows normalized 2-d histograms of samples projected onto a plane. The first column is the training data, column two is the Gibbs distribution, column three is a Gaussian distribution, and column four is a Product-of-Student-t distribution.

## 3   Results

We trained several models on several datasets. We show results for complete models of the joint neuronal response of 5 real motor cortex cells plus $x, y$ hand kinematics (3 past spikes for each cell plus 2 behavior variables equals a 17 dimension dataset). A complete model has the same number of experts as dimensions.

Table 1 shows the log likelihood of test data under several models: Product of Student-t, a diagonal covariance multidimensional Gaussian (independent), multivariate Gaussian, a complete FRAME model with random projection directions, a product of 5 complete FRAME single cell models with learned projections, and a complete FRAME model with learned projection directions. Because these all are complete models, we are able to compute the partition function of each. Each model was trained on 4000 points and the log likelihood was computed using 1000 distinct test points.

In Fig. 4 we show histograms of samples drawn from a full covariance Gaussian and energy-based models with two times more projection directions than the data dimensionality. These figures illustrate the modeling power of our approach in that it represents the irregularities common to real neural data better than Gaussian and other symmetric distributions.

Note that the model using random marginal directions does not model the data as well as one using optimized directions; this is not surprising. It may well be the case, however, that with many more random directions such a model would perform significantly better. This overcomplete case however is unnormalized and hence cannot be directly compared here.

# 4 Discussion

In this work we demonstrated an approach for using Gibbs distributions to model the joint spiking activity of a population of cells and an associated behavior. We developed a novel application of contrastive divergence for learning a FRAME model which can be viewed as a semi-parametric Product-of-Experts model. We showed that our model outperformed other models in representing complex monkey motor cortical spiking data.

Previous methods for probabilistically modeling spiking process have focused on modeling the firing rates of a population in terms of a conditional intensity function (firing rate conditioned on various correlates and previous spiking) [15, 16, 17, 18, 19]. These functions are often formulated in terms of log-linear models and hence resemble our approach. Here we take a more direct approach of modeling the joint probability using energy-based models and exploit contrastive divergence for learning

Information theoretic analysis of spiking populations calls for modeling high dimensional joint and conditional distributions. In the work of [20, 21, 22], these distributions are used to study encoding models, in particular the importance of correlation in the neural code. Our models are directly applicable to this pursuit. Given an experimental design with a relatively low dimension stimulus, where the entropy of that stimulus can be accurately computed, our models are applicable without modification.

Our approach may also be applied to neural decoding. A straightforward extension of our model could include hand positions (or other kinematic variables) at multiple time instants. Decoding algorithms that exploits these joint models by maximizing the likelihood of the observed firing activity over an entire data set remain to be developed. Note that it may be possible to produce more accurate models of the un-normalized joint probability by increasing the number of marginal constraints. To exploit these overcomplete models, algorithms that do not require normalized probabilities are required (particle filtering is a good example).

Not surprisingly the FRAME model performed better on the non-symmetric neural data than the related, but symmetric, Product-of-Student-t model. We have begun exploring more flexible and asymmetric experts which would offer advantages over discrete histogramming inherent to the FRAME model.

### Acknowledgments

Thanks to J. Donoghue, W. Truccolo, M. Fellows, and M. Serruya. This work was supported by NIH-NINDS R01 NS 50967-01 as part of the NSF/NIH Collaborative Research in Computational Neuroscience Program.

## References

[1] S. C. Zhu, Z. N. Wu, and D. Mumford, "Minimax entropy principle and its application to texture modeling," *Neural Comp.*, vol. 9, no. 8, pp. 1627–1660, 1997.

[2] C. Liu, S. C. Zhu, and H. Shum, "Learning inhomogeneous Gibbs model of faces by minimax entropy," in *ICCV*, pp. 281–287, 2001.

[3] G. Hinton, "Training products of experts by minimizing contrastive divergence," *Neural Comp.*, vol. 14, pp. 1771–1800, 2002.

[4] Y. Teh, M. Welling, S. Osindero, and G. E. Hinton, "Energy-based models for sparse overcomplete representations," *JMLR*, vol. 4, pp. 1235–1260, 2003.

[5] G. Hinton, "Product of experts," in *ICANN*, vol. 1, pp. 1–6, 1999.

[6] A. J. Bell and L. C. Parra, "Maximising sensitivity in a spiking network," in *Advances in NIPS*, vol. 17, pp. 121–128, 2005.

[7] R. S. Zemel, Q. J. M. Huys, R. Natarajan, and P. Dayan, "Probabilistic computation in spiking populations," in *Advances in NIPS*, vol. 17, pp. 1609–1616, 2005.

[8] M. Serruya, N. Hatsopoulos, M. Fellows, L. Paninski, and J. Donoghue, "Robustness of neuroprosthetic decoding algorithms," *Biological Cybernetics*, vol. 88, no. 3, pp. 201–209, 2003.

[9] M. D. Serruya, N. G. Hatsopoulos, L. Paninski, M. R. Fellows, and J. P. Donoghue, "Brain-machine interface: Instant neural control of a movement signal," *Nature*, vol. 416, pp. 141–142, 2002.

[10] W. Wu, M. J. Black, Y. Gao, E. Bienenstock, M. Serruya, A. Shaikhouni, and J. P. Donoghue, "Neural decoding of cursor motion using a Kalman filter," in *Advances in NIPS*, vol. 15, pp. 133–140, 2003.

[11] Y. Gao, M. J. Black, E. Bienenstock, S. Shoham, and J. P. Donoghue, "Probabilistic inference of arm motion from neural activity in motor cortex," *Advances in NIPS*, vol. 14, pp. 221–228, 2002.

[12] M. Welling, G. Hinton, and S. Osindero, "Learning sparse topographic representations with products of Student-t distributions," in *Advances in NIPS*, vol. 15, pp. 1359–1366, 2003.

[13] A. Gelman, J. B. Carlin, H. S. Stern, and D. B. Rubin, *Bayesian Data Analysis*, 2nd ed. Chapman & Hall/CRC, 2004.

[14] S. Roth and M. J. Black, "Fields of experts: A framework for learning image priors," in *CVPR*, vol. 2, pp. 860–867, 2005.

[15] D. R. Brillinger, "The identification of point process systems," *The Annals of Probability*, vol. 3, pp. 909–929, 1975.

[16] E. S. Chornoboy, L. P. Schramm, and A. F. Karr, "Maximum likelihood identification of neuronal point process systems," *Biological Cybernetics*, vol. 59, pp. 265–275, 1988.

[17] Y. Gao, M. J. Black, E. Bienenstock, W. Wu, and J. P. Donoghue, "A quantitative comparison of linear and non-linear models of motor cortical activity for the encoding and decoding of arm motions," in *First International IEEE/EMBS Conference on Neural Engineering*, pp. 189–192, 2003.

[18] M. Okatan, "Maximum likelihood identification of neuronal point process systems," *Biological Cybernetics*, vol. 59, pp. 265–275, 1988.

[19] W. Truccolo, U. T. Eden, M. R. Fellows, J. P. Donoghue, and E. N. Brown, "A point process framework for relating neural spiking activity to spiking history," *J. Neurophysiology*, vol. 93, pp. 1074–1089, 2005.

[20] P. E. Latham and S. Nirenberg, "Synergy, redundancy, and independence in population codes, revisited," *J. Neuroscience*, vol. 25, pp. 5195–5206, 2005.

[21] S. Nirenberg and P. E. Latham, "Decoding neuronal spike trains: How important are correlations?" *PNAS*, vol. 100, pp. 7348–7353, 2003.

[22] S. Panzeri, H. D. R. Golledge, F. Zheng, M. Tovee, and M. P. Young, "Objective assessment of the functional role of spike train correlations using information measures," *Visual Cognition*, vol. 8, pp. 531–547, 2001.
